# A Fast Multi-Resolution Method for Detection of Significant Spatial Disease Clusters

**Daniel B. Neill**
Department of Computer Science
Carnegie Mellon University
Pittsburgh, PA 15213
neill@cs.cmu.edu

**Andrew W. Moore**
Department of Computer Science
Carnegie Mellon University
Pittsburgh, PA 15213
awm@cs.cmu.edu

## Abstract

Given an $N \times N$ grid of squares, where each square has a *count* and an underlying *population*, our goal is to find the square region with the highest *density*, and to calculate its *significance* by randomization. Any *density measure D*, dependent on the total count and total population of a region, can be used. For example, if each count represents the number of disease cases occurring in that square, we can use Kulldorff's *spatial scan statistic $D_K$* to find the most significant spatial disease cluster. A naive approach to finding the maximum density region requires $O(N^3)$ time, and is generally computationally infeasible. We present a novel algorithm which partitions the grid into overlapping regions, bounds the maximum score of subregions contained in each region, and prunes regions which cannot contain the maximum density region. For sufficiently dense regions, this method finds the maximum density region in optimal $O(N^2)$ time, in practice resulting in significant (10-200x) speedups.

## 1 Introduction

This paper develops fast methods for *detection of spatial overdensities*: discovery of spatial regions with high scores according to some density measure, and statistical significance testing in order to determine whether these high-density regions can reasonably have occurred by chance. A major application is in identifying clusters of disease cases, for purposes ranging from detection of bioterrorism (ex. anthrax) to environmental risk factors for diseases such as childhood leukemia ([1]-[3]). [4] discusses many other applications, including astronomy (identifying star clusters), reconnaissance, and medical imaging.

Consider the case in which counts are aggregated to a uniform 2-d grid. Assume an $N \times N$ grid of squares $G$, where each square $s_{ij} \in G$ is associated with a *count $c_{ij}$* and an underlying *population $p_{ij}$*. For example, a square's count may be the number of disease cases in that geographical region in a given time period, while its population may be the total number of people "at-risk" for the disease. Our goal is to find the square region $S^* \subseteq G$ with the highest *density* according to a density measure $D$: $S^* = \arg\max_S D(S)$. We use the abbreviations *mdr* for the "maximum density region" $S^*$, and *mrd* for the "maximum region density" $D(S^*)$, throughout. The density measure $D$ must be an increasing function of the total count of the region, $C(S) = \sum_S c_{ij}$, and a decreasing function of the total population of the region, $P(S) = \sum_S p_{ij}$. In the case of a uniform underlying population, $P(S) \propto k^2$, where $k$ is the size of region $S$. But we focus on the more interesting case: non-uniform populations.

The problem of finding significant spatial overdensities is distinct from that solved by grid-based hierarchical methods such as CLIQUE [5], MAFIA [6], and STING [7], which also look for "dense clusters." There are three main differences:

1. Our method is applicable to any density measure $D$, while the other algorithms are specific to the "standard" density measure $D_1(S) = \frac{C(S)}{P(S)}$. The $D_1$ measure is the number of points per unit population, for example this corresponds to the region with the highest observed disease rate. Unlike many other density measures, $D_1$ is *monotonic*: if a region $S$ with density $d$ is partitioned into any set of disjoint subregions, at least one subregion will have density $d' \geq d$. Thus it is not particularly useful to find the "region" with maximum $D_1$, since this will be the single square with highest $\frac{c_{ij}}{p_{ij}}$. Instead, the other algorithms search for maximally sized regions with $D_1$ greater than some threshold, relying on the monotonicity of $D_1$ by first finding dense units ($1 \times 1$ squares), then merging adjacent units in bottom-up fashion. For a non-monotonic measure such as Kulldorff's, it is possible to have a large dense region where none of its subregions are themselves dense, so bottom-up can fail. Here, we will optimize with respect to arbitrary non-monotonic density measures, and thus use a different approach from CLIQUE, MAFIA, or STING.

2. Our method deals with non-uniform underlying populations: this is particularly important for real-world epidemiological applications, in which an overdensity of disease cases is more significant if the underlying population is large.

3. Our goal is not only to find the highest scoring region, but also to test whether that region is a true cluster or if it is likely to have occurred by chance.

## 1.1   The spatial scan statistic

A non-monotonic density measure which is of great interest to epidemiologists is Kulldorff's *spatial scan statistic* [8], which we denote by $D_K$. This assumes that counts $c_{ij}$ are generated by an inhomogeneous Poisson process with mean $qp_{ij}$, where $q$ is the underlying "disease rate" (or expected value of the $D_1$ density). We then calculate the log of the likelihood ratio of two possibilities: that the disease rate $q$ is higher in the region than outside the region, and that the disease rate is identical inside and outside the region. For a region with count $C$ and population $P$, in a grid with total count $C_{tot}$ and population $P_{tot}$, we can calculate $D_K = C \log \frac{C}{P} + (C_{tot} - C) \log \frac{C_{tot} - C}{P_{tot} - P} - C_{tot} \log \frac{C_{tot}}{P_{tot}}$, if $\frac{C}{P} > \frac{C_{tot}}{P_{tot}}$, and 0 otherwise. [8] proved that the spatial scan statistic is *individually most powerful* for finding a significant region of elevated disease rate: it is more likely to detect the overdensity than any other test statistic. Note, however, that our algorithm is general enough to use any density measure, and in some cases we may wish to use measures other than Kulldorff's. For instance, if we have some idea of the size of the maximum density region, we can use the $D_r$ measure, $D_r(S) = \frac{C(S)}{P(S)^r}$, $0 < r < 1$, with larger $r$ corresponding to tests for smaller clusters.

Once we have found the maximum density region (mdr) of grid $G$ according to our density measure, we must still determine the statistical significance of this region. Since the exact distribution of the test statistic is only known in special cases (such as $D_1$ density with a uniform underlying population), in general we must perform Monte Carlo simulation for our hypothesis test. To do so, we run a large number $R$ of random replications, where a replica has the same underlying populations $p_{ij}$ as $G$, but assumes a uniform disease rate $q_{rep} = \frac{C_{tot}(G)}{P_{tot}(G)}$ for all squares. For each replica $G'$, we first generate all counts $c_{ij}$ randomly from an inhomogeneous Poisson distribution with mean $q_{rep}p_{ij}$, then compute the maximum region density (mrd) of $G'$ and compare this to mrd($G$). The number of replicas $G'$ with mrd($G'$) $\geq$ mrd($G$), divided by the total number of replications $R$, gives us the *p*-value for our maximum density region. If this *p*-value is less than .05, we can conclude that the discovered region is statistically significant (unlikely to have occurred

by chance) and is thus a "spatial overdensity." If the test fails, we have still discovered the maximum density region of $G$, but there is not sufficient evidence that this is an overdensity.

## 1.2 The naive approach

The simplest method of finding the maximum density region is to compute the density of all square regions of sizes $k = k_{min} \ldots N$.[1] Since there are $(N - k + 1)^2$ regions of size $k$, there are a total of $O(N^3)$ regions to examine. We can compute the density of any region $S$ in $O(1)$, by first finding the count $C(S)$ and population $P(S)$, then applying our density measure $D(C, P)$.[2] This allows us to compute the mdr of an $N \times N$ grid $G$ in $O(N^3)$ time. However, significance testing by Monte Carlo replication also requires us to find the mrd for each replica $G'$, and compare this to mrd$(G)$. Since calculation of the mrd takes $O(N^3)$ time for each replica, the total complexity is $O(RN^3)$, and $R$ is typically large (we assume $R = 1000$). Several simple tricks may be used to speed up this procedure for cases where there is no significant spatial overdensity. First, we can stop examining a replica $G'$ immediately if we find a region with density greater than mrd$(G)$. Second, we can use the Central Limit Theorem to halt our Monte Carlo testing early if, after a number of replications $R' < R$, we can conclude with high confidence that the region is not significant. For cases where there *is* a significant spatial overdensity, the naive approach is still extremely computationally expensive, and this motivates our search for a faster algorithm.

## 2 Overlap-multires partitioning

Since the problem of detection of spatial overdensities is closely related to problems such as kernel density estimation and kernel regression, this suggests that multi-resolution partitioning techniques such as kd-trees [9] and mrkd-trees [10] may be useful in speeding up our search. The main difference of our problem from kernel density estimation, however, is that we are only interested in the maximum density region; thus, we do not necessarily need to build a space-partitioning tree at all resolutions. Also, the assumption that counts are aggregated to a uniform grid simplifies and speeds up partitioning, eliminating the need for a computationally expensive instance-based approach. These observations suggest a top-down multi-resolution partitioning approach, in which we search first at coarse resolutions (large regions), then at successively finer resolutions as necessary. One option would be to use a "quadtree" [11], a hierarchical data structure in which each region is recursively partitioned into its top left, top right, bottom left, and bottom right quarters. However, a simple partitioning approach fails because of the non-monotonicity of our density measure: a dense region may be split into two or more separate subregions, none of which is as dense as the original region. This problem can be prevented by a partitioning approach in which adjacent regions partially overlap, an approach we call "overlap-multires partitioning."

To explain how this method works, we first define some notation. We denote a region $S$ by an ordered triple $(x, y, k)$, where $(x, y)$ is the upper left corner of the region and $k$ is its size. Next, we define the $\omega$-children of a region $S = (x, y, k)$ as the four overlapping subregions of size $k - \omega$ corresponding to the top left, top right, bottom left, and bottom right corners of $S$: $(x, y, k - \omega)$, $(x + \omega, y, k - \omega)$, $(x, y + \omega, k - \omega)$, and $(x + \omega, y + \omega, k - \omega)$. Next, we define a region as "even" if its size is $2^k$ for some $k \geq 2$, and "odd" if its size is $3 \times 2^k$ for some $k \geq 0$. We define the "gridded children" (g-children) of an even region $S = (x, y, k)$ as its $\omega$-children for $\omega = \frac{k}{4}$. Thus the four g-children of an even region are odd, and each overlaps $\frac{2}{3}$ with the directly adjacent child regions. Similarly, we define the g-children of an odd region $S = (x, y, k)$ as its $\omega$-children for $\omega = \frac{k}{3}$. Thus the four g-children of an odd region are even, and each overlaps $\frac{1}{2}$ with the directly adjacent child regions. Note that

even though a region has four g-children, and each of its g-children has four g-children, it has only nine (not 16) distinct grandchildren, several of which are the child of multiple regions. Figure 1 shows the first two levels of such a tree.

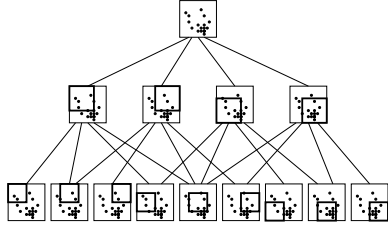

Figure 1: The first two levels of the overlap-mutires tree. Each node represents a gridded region (denoted by a thick square) of the entire dataset (thin square and dots).

Next, we assume that the size of the entire grid is a power of two: thus the entire grid $G = (0,0,N)$ is an even region. We define the set of "gridded" regions of $G$ as $G$ and all of its "gridded descendents" (its g-children, g-grandchildren, etc.). Our algorithm focuses its search on the set of gridded regions, only searching non-gridded regions when necessary. This technique is useful because the total number of gridded regions is $O(N^2)$, as in the simple quadtree partitioning method. This implies that, if only gridded regions need to be searched, our total time to find the mdr of a grid is $O(N^2)$. Since it takes $\Omega(N^2)$ time to generate the grid, this time bound is optimal.

## 2.1 Top-down pruning

So when can we search only gridded regions, or alternatively, when does a given non-gridded region need to be searched? Our basic method is branch-and-bound: we perform a top-down search, and speed up this search by *pruning* regions which cannot possibly contain the mdr. Our first step is to derive an upper bound $D_{max}(S,k)$ on the density of subregions of minimum size $k$ contained in a given region $S$ (Section 2.2). Then we can compare $D_{max}(S,k)$ to the density $D(S^*)$ of the best region found so far: if $D_{max}(S,k) < D(S^*)$, we know that no subregion of $S$ with size $k$ or more can be the mdr.

We can use this information for two types of pruning. First, if $D_{max}(S,k_{min}) < D(S^*)$, we know that *no* subregion of $S$ can be optimal; we can prune the region completely, and not search its (gridded or non-gridded) children. Second, we can show that (for $0 < k < n$) any region of size $2^k + 1$ or less is contained entirely in an odd gridded region of size $\frac{3}{2} \times 2^k$. Thus, if $D_{max}(G, 2^{n-1} + 2) < D(S^*)$ for the entire grid $G$, any optimal non-gridded region must be contained in an odd gridded region. Similarly, if $D_{max}(S, 2^k + 2) < D(S^*)$ for an odd gridded region $S$ of size $3 \times 2^k$, any optimal non-gridded subregion of $S$ must be within an odd gridded subregion of $S$. Thus we can search only gridded regions if two conditions hold: 1) no subregion of $G$ of size $2^{n-1} + 2$ or more can be optimal, and 2) for each odd gridded region of size $3 \times 2^k$, no subregion of size $2^k + 2$ or more can be optimal.

## 2.2 Bounding subregion density

To bound the maximum subregion density $D_{max}(S,k)$, we must find the highest possible score $D(S')$ of a subregion $S' \subseteq S$ of size $k$ or more. Let $C = C(S)$, $P = P(S)$, and $K = \text{size}(S)$. We assume that these are known, as well as lower and upper bounds $[d_{min}, d_{max}]$ on the $D_1$ density of subregions of $S$. Let $c = C(S')$ and $p = P(S')$; these are presently unknown. We can prove that, if $D(S') > D(S)$, the maximum value of $D(S')$ occurs when $S$ has the maximum allowable $D_1$ density $d_{max}$, and $S - S'$ has the minimum allowable $D_1$ density $d_{min}$: this gives us $pd_{max} + (P - p)d_{min} = C$. Thus $p = \frac{C - Pd_{min}}{d_{max} - d_{min}}$ and $c = d_{max}p = \frac{C - Pd_{min}}{1 - d_{min}/d_{max}}$. Then computing $D(c,p)$ gives us a guaranteed upper bound on $D_{max}(S,k)$.

We can place tighter bounds on $D_{max}(S,k)$ if we also have a lower bound $p_{min}(S,k)$ on the population of a size $k$ subregion $S' \subseteq S$: in this case, if the value calculated for $p$ in the

equation above is less than $p_{min}$, we know that $D(c', p_{min})$, where $c' = C - (P - p_{min})d_{min}$, is a tighter upper bound for $D_{max}$. We can bound $p_{min}$ in several ways. First, if we know the minimum population $p_{s,min}$ of a single square $s \in S$, then $p_{min} \geq k^2 p_{s,min}$. Second, if we know the maximum population $p_{s,max}$ of a single square $s \in S$, then $p_{min} \geq P - (K^2 - k^2)p_{s,max}$. At the beginning of our algorithm, we calculate $p_{s,max}(S) = \max p_{ij}$ and $p_{s,min}(S) = \min p_{ij}$ (where $s_{ij} \in S$) for each gridded region $S$. This calculation can be done recursively (bottom-up) in $O(N^2)$. The resulting population statistics are used for the original grid and for all replicas. For non-gridded regions, we use the population statistics of the region's gridded parent (either an odd gridded region or the entire grid $G$); these bounds will be looser for the child region than for the parent, but are still correct. We also initially calculate $d_{max}$ and $d_{min}$. This is done simply by finding the global maximum and minimum values of the $D_1$ density: $d_{max} = \max \frac{C(S')}{P(S')}$ (where $S' \subseteq G$ and size$(S') = k_{min}$), and $d_{min} = \min \frac{c_{ij}}{p_{ij}}$ (where $s_{ij} \in G$).[3] Alternatively, we could compute $d_{max}$ and $d_{min}$ recursively (bottom-up) for each gridded region $S$, but in practice we find that the global values are sufficient for good performance on most test cases.

## 2.3 The algorithm

Our algorithm, based on the overlap-multires partitioning scheme above, is a top-down, best-first search of the set of gridded regions, followed by a top-down, best-first search of any non-gridded regions as necessary. We use priority queues (q1,q2) for our search: each step of the algorithm takes the "best" (i.e. highest density) region from a queue, examines it, and (if necessary) adds its children to queues. The $\omega$-children and g-children of a region $S$ are defined above; note that the 1-children of $S$ are its $\omega$-children with $\omega = 1$. We also assume that regions are "marked" once added to a queue, so that a region will not be searched more than once. Finally, we use the rules and density bounds derived above to speed up our search, by pruning subregions when $D_{max}(S, k) \leq D(S^*)$. The basic pseudocode outline of our method is as follows:

```
Add G to q1.
If D_max(G,N/2+2)>mrd, add 1-children(G) to q2 with k1=N/2+2.
While q1 not empty:
  Get best region S from q1.
  If D(S)>mrd, set mdr=S and mrd=D(S).
  If D_max(S,k_min)>mrd, add g-children(S) to q1.
  If size(S)=3(2^k) and D_max(S,2^k+2)>mrd, add 1-children(S) to q2 with k1=2^k+2.
While q2 not empty:
  Get best region S and value k1(S) from q2.
  If D(S)>mrd, set mdr=S and mrd=D(S).
  If D_max(S,k1(S))>mrd, add 1-children(S) to q2 with same k1.
```

These steps are first performed for the original grid, allowing us to calculate its mdr and mrd. We then perform these steps to calculate the mrd of each replica; however, several techniques allow us to reduce the amount of computation necessary for a replica. First, we can stop examining a replica $G'$ immediately if we find a region with density greater than mrd$(G)$. This is especially useful in cases where there is no significant spatial overdensity in $G$. Second, we can use mrd$(G)$ for pruning our search on a replica $G'$: if $D_{max}(S, k) < $ mrd$(G)$ for some $S \subseteq G'$, we know that no subregion of $S$ of size $k$ or more can have a greater density than the mdr of the original grid, and thus we do not need to examine any of those subregions. This is especially useful where there is a significant spatial overdensity in $G$: a high mrd will allow large amounts of pruning on the replica grids.

## 3  Improving the algorithm

The exact version of the algorithm uses conservative estimates of the $D_1$ densities of $S'$ and $S - S'$ ($d_{max}$ and $d_{min}$ respectively), and a loose lower bound on the population of $S'$, to

calculate $D_{max}(S,k)$. This results in a loose upper bound on $D_{max}$ which is guaranteed to be correct, but allows little pruning to be done. We can derive tighter bounds on $D_{max}$ in two ways: by using a closer approximation to the $D_1$ density of $S - S'$, and by using a tighter lower bound on the population of $S'$. These improvements are discussed below.

## 3.1 The outer density approximation

To derive tighter bounds on the maximum density of a subregion $S'$ contained in a given region $S$, we first note that (under both the null hypothesis and the alternative hypothesis) we assume that at most one disease cluster $S_{dc}$ exists, and that the disease rate $q$ is expected to be uniform outside $S_{dc}$ (or uniform everywhere, if no disease cluster exists). Thus, if $S_{dc}$ is contained entirely in the region under consideration $S$, we would expect that the maximum density subregion $S'$ of $S$ is $S_{dc}$, and that the disease rate of $S - S'$ is equal to the disease rate outside $S$: $E\left[\frac{C-c}{P-p}\right] = \frac{C_{tot}-C}{P_{tot}-P} = d_{out}$. Assuming that the $D_1$ density of $S - S'$ is equal to its expected value $d_{out}$, we obtain the equation $pd_{max} + (P-p)d_{out} = C$. Solving for $p$, we find $p = \frac{C-Pd_{out}}{d_{max}-d_{out}}$. Then $D_{max}(S,k) = D(c,p)$, where $c = d_{max}p$.

The problem with this approach is that we have not compensated for the variance in densities: our calculated value of $D_{max}$ is an upper bound for the maximum subregion density $D(S')$ only in the most approximate probabilistic sense. We would expect the $D_1$ density of $S - S'$ to be less than its expected value half the time, and thus we would expect $D(S')$ to be less than $D_{max}$ at least half the time; in practice, our bound will be correct more often, since we are still using a conservative approximation of the $D_1$ density of $S'$. Note also that we expect to underestimate $D_{max}$ if the disease cluster $S_{dc}$ is not contained entirely in $S$: this is acceptable (and desirable) since a region not containing $S_{dc}$ does not need to be expanded.

We can improve the correctness of our probabilistic bound by also considering the variance of $\frac{C-c}{P-p} - \frac{C_{tot}-C}{P_{tot}-P}$. Assuming that all counts outside $S_{dc}$ are generated by a inhomogeneous Poisson distribution with parameter $qp_{ij}$, we obtain: $\sigma^2\left[\frac{C-c}{P-p} - \frac{C_{tot}-C}{P_{tot}-P}\right] = \sigma^2\left[\frac{\text{Po}(q(P-p))}{P-p} - \frac{\text{Po}(q(P_{tot}-P))}{P_{tot}-P}\right] = \frac{q}{P-p} + \frac{q}{P_{tot}-P} = \frac{q(P_{tot}-p)}{(P-p)(P_{tot}-P)}$. Since the actual value of the parameter $q$ is not known, we use a conservative empirical estimate: $q = \frac{C_{tot}}{P_{tot}-p}$. From this, we obtain $\sigma\left[\frac{C-c}{P-p} - \frac{C_{tot}-C}{P_{tot}-P}\right] = \sqrt{\frac{C_{tot}}{(P-p)(P_{tot}-P)}}$. Then we can compute $p$ by solving $pd_{max} + (P-p)(d_{out} - b\sigma) = C$, and obtain $c = d_{max}p$ and $D_{max} = D(c,p)$ as before.

By adjusting our approximation of the minimum density in this manner, we compute a higher score $D_{max}$, reducing the likelihood that we will underestimate the maximum subregion density and prune a region that should not necessarily be pruned. Given a constant $b$, the $D_1$ density of $S - S'$ will be greater than $d_{out} - b\sigma$ with probability $P(Z < b)$, where $Z$ is chosen randomly from the unit normal. For $b = 2$, there is an 98% chance that we will underestimate $D_1(S - S')$, giving a guaranteed correct upper bound for the maximum subregion density. In practice, the maximum subregion density will be lower than our computed value of $D_{max}$ more often, since our estimates for $d_{max}$ and $q$ are conservative. Thus, though our algorithm is approximate, it is very likely to converge to the globally optimal mdr. In fact, our experiments demonstrate that $b = 1$ is sufficient to obtain the correct region with over 90% probability, approaching 100% for sufficiently dense regions.

## 3.2 Cached population statistics

A final step in making the algorithm tractable is to cache certain statistics about the minimum populations of subsquares of gridded regions. This is only performed once: it need not be repeated for each replica (since populations need not be randomized). Although there is no room to describe it, we have empirically shown it to give an important acceleration if populations are highly non-uniform. The results below make use of this.

# 4 Results

We first describe results with artificially generated grids and then real-world case data. An artificial grid is generated from a set of parameters ($N$, $k$, $\mu$, $\sigma$, $q'$, $q''$). The grid generator first creates an $N \times N$ grid, and randomly selects a $k \times k$ "test region." Then the population of each square is chosen randomly from a normal distribution with mean $\mu$ and standard deviation $\sigma$ (populations less than zero are set to zero). Finally, the count of each square is chosen randomly from a Poisson distribution with parameter $qp_{ij}$, where $q = q'$ inside the test region and $q = q''$ outside the test region.

We tested three different adjustments for density variance ($b = 0, 1, 2$). The approximate algorithm was tested for grids of size $N = 512$; test region sizes of $k = 16$ and $k = 4$ were used, and the disease rate $q$ was set to .002 inside the test region and .001 outside the test region. We used three different population distributions for testing: the "standard" distribution ($\mu = 10^4$, $\sigma = 10^3$), and two types of "highly varying" populations. For the "city" distribution, we randomly selected a "city region" with size 16: square populations were generated with $\mu = 10^7$ and $\sigma = 10^6$ inside the city, and $\mu = 10^4$ and $\sigma = 10^3$ outside the city. For the "high-$\sigma$" distribution, we generated all square populations with $\mu = 10^4$ and $\sigma = 5 \times 10^3$. We first compared the performance of each variant of the algorithm to the naive approach for the three test cases; see Table 1 for results. For large test regions ($k = 16$), all variants of the algorithm had runtimes of ~20 minutes, as compared to 44 hours for the naive approach, a speedup of 122-155x. For small test regions ($k = 4$), we observed that performance generally decreased with increasing $b$: the algorithm achieved average speedups of 133x for $b = 0$, 61x for $b = 1$, and 18x for $b = 2$.

Next, we tested accuracy by generating 50 artificial grids for each population distribution, and computing the percentage of test grids on which the algorithm was able to find the correct mdr (see Table 2). For the large test region ($k = 16$), all variants were able to find the correct mdr with high (97-100%) accuracy. For the small test region, accuracy improved significantly with increasing $b$: the non-variance adjusted version ($b = 0$) achieved only 45% accuracy, while the variance adjusted versions ($b = 1$ and $b = 2$) achieved 89% and 99% accuracy respectively. These results demonstrate that the approximate algorithm (with variance adjustment and cached population statistics) is able to achieve high performance and accuracy even for very small test regions and highly non-uniform populations.

Finally, we measured the performance of the approximate algorithm on a grid generated from real-world data. We used a database of (anonymized) Emergency Department data collected from Western Pennsylvania hospitals in the period 1999-2002. This dataset contained a total of 630,000 records, each representing a single ED visit and giving the latitude and longitude of the patient's home location to the nearest .005 degrees ($\sim \frac{1}{3}$ mile, a sufficiently low resolution to ensure anonymity). For each record, the latitude $L$ and longitude $l$ were converted to a grid square $s_{ij}$ by $i = \frac{L - L_{min}}{.005}$ and $j = \frac{l - l_{min}}{.005}$; this created a $512 \times 512$ grid. We tested for spatial clustering of "recent" disease cases: the "count" of each square was the number of ED visits in that square in the last two months, and the "population" of that square was the total number of ED visits in that square. See Figure 2 for a picture of this dataset, including the highest scoring region. We tested six variants of the approximate algorithm on the ED dataset; the presence/absence of cached population statistics did not significantly affect the performance or accuracy for this test, so we focus on the variation in $b$. All three variants ($b = 0, 1, 2$), as well as the naive algorithm, found the maximum density region (of size 101) and found it statistically significant ($p$-value 0/1000). The major difference, of course, was in runtime and number of regions searched (see Table 3). The naive algorithm took 2.7 days to find the mdr and perform 1000 Monte Carlo replications, while each of the variants of the approximate algorithm performed the same task in ~2 hours or less. The approximate algorithm took 19 minutes (a speedup of 209x) for $b = 0$, 47 minutes (a speedup of 85x) for $b = 1$, and 126 minutes (a speedup of 31x) for $b = 2$. Thus we can see that all three variants find the correct region in much less time than

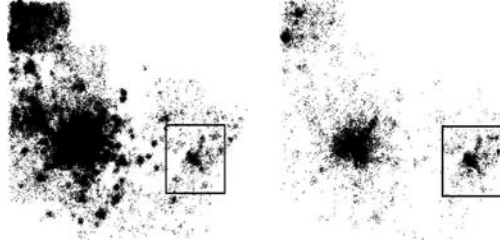

Figure 2: The left picture shows the "population" distribution within Western PA and the right picture shows the "counts" distribution. The winning region is shown as a square.

Table 1: Performance of algorithm, $N = 512$

| method | test | time (orig+1000 reps) | speedup |
|---|---|---|---|
| naive | all | $2:37+43:36:40$ | x1 |
| $b=0$ | std, $k=16$ | $0:42+16:40$ | x151 |
| $b=1$ | std, $k=16$ | $0:43+16:20$ | x154 |
| $b=2$ | std, $k=16$ | $0:41+17:00$ | x148 |
| $b=0$ | std, $k=4$ | $0:41+17:00$ | x148 |
| $b=1$ | std, $k=4$ | $0:41+29:10$ | x88 |
| $b=2$ | std, $k=4$ | $0:42+1:13:00$ | x36 |
| $b=0$ | city, $k=16$ | $0:42+16:30$ | x153 |
| $b=1$ | city, $k=16$ | $0:46+20:40$ | x122 |
| $b=2$ | city, $k=16$ | $0:41+18:40$ | x135 |
| $b=0$ | city, $k=4$ | $0:43+24:30$ | x104 |
| $b=1$ | city, $k=4$ | $0:44+2:11:00$ | x20 |
| $b=2$ | city, $k=4$ | $0:47+7:06:50$ | x6.1 |
| $b=0$ | high-$\sigma$, $k=16$ | $0:41+17:00$ | x148 |
| $b=1$ | high-$\sigma$, $k=16$ | $0:41+16:40$ | x151 |
| $b=2$ | high-$\sigma$, $k=16$ | $0:41+17:00$ | x148 |
| $b=0$ | high-$\sigma$, $k=4$ | $0:44+17:15$ | x146 |
| $b=1$ | high-$\sigma$, $k=4$ | $0:45+34:10$ | x75 |
| $b=2$ | high-$\sigma$, $k=4$ | $1:08+3:20:00$ | x13 |

Table 2: Accuracy of algorithm

| method | test | accuracy ($k=16$) | accuracy ($k=4$) |
|---|---|---|---|
| $b=0$ | standard | 96% | 52% |
| $b=0$ | city | 98% | 36% |
| $b=0$ | high-$\sigma$ | 98% | 46% |
| $b=1$ | standard | 100% | 90% |
| $b=1$ | city | 100% | 88% |
| $b=1$ | high-$\sigma$ | 100% | 90% |
| $b=2$ | standard | 100% | 98% |
| $b=2$ | city | 100% | 98% |
| $b=2$ | high-$\sigma$ | 100% | 100% |

Table 3: Emergency Dept. dataset

| method | time (orig+1000 reps) | speedup |
|---|---|---|
| naive | $4:05+65:50:00$ | x1 |
| $b=0$ | $4:20+14:36$ | x209 |
| $b=1$ | $4:22+42:20$ | x85 |
| $b=2$ | $4:36+2:01:12$ | x31 |

the naive approach. This is very important for applications such as real-time detection of disease outbreaks: if a system is able to detect an outbreak in minutes rather than days, preventive measures or treatments can be administered earlier, possibly saving many lives.

Thus we have presented a fast overlap-multires partitioning algorithm for detection of spatial overdensities, and demonstrated that this method results in significant (10-200x) speedups on real and artificially generated datasets. We are currently applying this algorithm to national-level hospital and pharmacy data, attempting to detect statistically significant indications of a disease outbreak based on changes in the spatial clustering of disease cases. Application of a fast partitioning method using the techniques presented here may allow us to achieve the difficult goal of automatic real-time detection of disease outbreaks.

## Footnotes

[1]We assume that a region must have size at least $k_{min}$ to be significant: here $k_{min} = 3$.

[2]An old trick allows us to compute the count of any $k \times k$ region in $O(1)$: we first form a matrix of the cumulative counts, then compute each region's count by adding at most four cumulative counts.

[3]We can use the tighter bound for $d_{max}$ since we are using it to bound the density of a square region $S'$ of size at least $k_{min}$; we cannot use the tighter bound for $d_{min}$ since $S - S'$ is not square.

# References

[1] S. Openshaw, et al. 1988. Investigation of leukemia clusters by use of a geographical analysis machine. *Lancet* **1**, 272-273.

[2] L. A. Waller, et al. 1994. Spatial analysis to detect disease clusters. In N. Lange, ed. *Case Studies in Biometry*. Wiley, 3-23.

[3] M. Kulldorff and N. Nagarwalla. 1995. Spatial disease clusters: detection and inference. *Statistics in Medicine* **14**, 799-810.

[4] M. Kulldorff. 1999. Spatial scan statistics: models, calculations, and applications. In Glaz and Balakrishnan, eds. *Scan Statistics and Applications*. Birkhauser: Boston, 303-322.

[5] R. Agrawal, et al. 1998. Automatic subspace clustering of high dimensional data for data mining applications. *Proc. ACM-SIGMOD Intl. Conference on Management of Data*, 94-105.

[6] S. Goil, et al. 1999. MAFIA: efficient and scalable subspace clustering for very large data sets. *Northwestern University, Technical Report No. CPDC-TR-9906-010.*

[7] W. Wang, et al. 1997. STING: a statistical information grid approach to spatial data mining. *Proc. 23rd Conference on Very Large Databases*, 186-195.

[8] M. Kulldorff. 1997. A spatial scan statistic. *Communications in Statistics: Theory and Methods* **26**(6), 1481-1496.

[9] F. P. Preparata and M. I. Shamos. 1985. *Computational Geometry: An Introduction.* Springer-Verlag: New York.

[10] K. Deng and A. W. Moore. 1995. Multiresolution instance-based learning. *Proc. 12th Intl. Joint Conference on Artificial Intelligence*, 1233-1239.

[11] H. Samet. 1990. *The Design and Analysis of Spatial Data Structures.* Addison-Wesley: Reading.
